# Ensemble Nyström Method

**Sanjiv Kumar**
Google Research
New York, NY
sanjivk@google.com

**Mehryar Mohri**
Courant Institute and Google Research
New York, NY
mohri@cs.nyu.edu

**Ameet Talwalkar**
Courant Institute of Mathematical Sciences
New York, NY
ameet@cs.nyu.edu

## Abstract

A crucial technique for scaling kernel methods to very large data sets reaching or exceeding millions of instances is based on low-rank approximation of kernel matrices. We introduce a new family of algorithms based on mixtures of Nyström approximations, *ensemble Nyström algorithms*, that yield more accurate low-rank approximations than the standard Nyström method. We give a detailed study of variants of these algorithms based on simple averaging, an exponential weight method, or regression-based methods. We also present a theoretical analysis of these algorithms, including novel error bounds guaranteeing a better convergence rate than the standard Nyström method. Finally, we report results of extensive experiments with several data sets containing up to 1M points demonstrating the significant improvement over the standard Nyström approximation.

## 1 Introduction

Modern learning problems in computer vision, natural language processing, computational biology, and other areas are often based on large data sets of tens of thousands to millions of training instances. But, several standard learning algorithms such as support vector machines (SVMs) [2, 4], kernel ridge regression (KRR) [14], kernel principal component analysis (KPCA) [15], manifold learning [13], or other kernel-based algorithms do not scale to such orders of magnitude. Even the storage of the kernel matrix is an issue at this scale since it is often not sparse and the number of entries is extremely large. One solution to deal with such large data sets is to use an approximation of the kernel matrix. As shown by [18], later by [6, 17, 19], low-rank approximations of the kernel matrix using the Nyström method can provide an effective technique for tackling large-scale scale data sets with no significant decrease in performance.

This paper deals with very large-scale applications where the sample size can reach millions of instances. This motivates our search for further improved low-rank approximations that can scale to such orders of magnitude and generate accurate approximations. We show that a new family of algorithms based on mixtures of Nyström approximations, *ensemble Nyström algorithms*, yields more accurate low-rank approximations than the standard Nyström method. Moreover, these ensemble algorithms naturally fit distributed computing environment where their computational cost is roughly the same as that of the standard Nyström method. This issue is of great practical significance given the prevalence of distributed computing frameworks to handle large-scale learning problems.

The remainder of this paper is organized as follows. Section 2 gives an overview of the Nyström low-rank approximation method and describes our ensemble Nyström algorithms. We describe several variants of these algorithms, including one based on simple averaging of $p$ Nyström solutions,

an exponential weight method, and a regression method which consists of estimating the mixture parameters of the ensemble using a few columns sampled from the matrix. In Section 3, we present a theoretical analysis of ensemble Nyström algorithms, namely bounds on the reconstruction error for both the Frobenius norm and the spectral norm. These novel generalization bounds guarantee a better convergence rate for these algorithms in comparison to the standard Nyström method. Section 4 reports the results of extensive experiments with these algorithms on several data sets containing up to 1M points, comparing different variants of our ensemble Nyström algorithms and demonstrating the performance improvements gained over the standard Nyström method.

## 2 Algorithm

We first give a brief overview of the Nyström low-rank approximation method, introduce the notation used in the following sections, and then describe our ensemble Nyström algorithms.

### 2.1 Standard Nyström method

We adopt a notation similar to that of [5, 9] and other previous work. The Nyström approximation of a symmetric positive semidefinite (SPSD) matrix $\mathbf{K}$ is based on a sample of $m \ll n$ columns of $\mathbf{K}$ [5, 18]. Let $\mathbf{C}$ denote the $n \times m$ matrix formed by these columns and $\mathbf{W}$ the $m \times m$ matrix consisting of the intersection of these $m$ columns with the corresponding $m$ rows of $\mathbf{K}$. The columns and rows of $\mathbf{K}$ can be rearranged based on this sampling so that $\mathbf{K}$ and $\mathbf{C}$ be written as follows:

$$\mathbf{K} = \begin{bmatrix} \mathbf{W} & \mathbf{K}_{21}^\top \\ \mathbf{K}_{21} & \mathbf{K}_{22} \end{bmatrix} \quad \text{and} \quad \mathbf{C} = \begin{bmatrix} \mathbf{W} \\ \mathbf{K}_{21} \end{bmatrix}. \tag{1}$$

Note that $\mathbf{W}$ is also SPSD since $\mathbf{K}$ is SPSD. For a uniform sampling of the columns, the Nyström method generates a rank-$k$ approximation $\widetilde{\mathbf{K}}$ of $\mathbf{K}$ for $k \leq m$ defined by:

$$\widetilde{\mathbf{K}} = \mathbf{C}\mathbf{W}_k^+ \mathbf{C}^\top \approx \mathbf{K}, \tag{2}$$

where $\mathbf{W}_k$ is the best $k$-rank approximation of $\mathbf{W}$ for the Frobenius norm, that is $\mathbf{W}_k = \text{argmin}_{\text{rank}(\mathbf{V})=k} \|\mathbf{W} - \mathbf{V}\|_F$ and $\mathbf{W}_k^+$ denotes the pseudo-inverse of $\mathbf{W}_k$ [7]. $\mathbf{W}_k^+$ can be derived from the singular value decomposition (SVD) of $\mathbf{W}$, $\mathbf{W} = \mathbf{U}\boldsymbol{\Sigma}\mathbf{U}^\top$, where $\mathbf{U}$ is orthonormal and $\boldsymbol{\Sigma} = \text{diag}(\sigma_1, \ldots, \sigma_m)$ is a real diagonal matrix with $\sigma_1 \geq \cdots \geq \sigma_m \geq 0$. For $k \leq \text{rank}(\mathbf{W})$, it is given by $\mathbf{W}_k^+ = \sum_{i=1}^k \sigma_i^{-1} \mathbf{U}^i {\mathbf{U}^i}^\top$, where $\mathbf{U}^i$ denotes the $i$th column of $\mathbf{U}$. Since the running time complexity of SVD is $O(m^3)$ and $O(nmk)$ is required for multiplication with $\mathbf{C}$, the total complexity of the Nyström approximation computation is $O(m^3 + nmk)$.

### 2.2 Ensemble Nyström algorithm

The main idea behind our ensemble Nyström algorithm is to treat each approximation generated by the Nyström method for a sample of $m$ columns as an *expert* and to combine $p \geq 1$ such experts to derive an improved hypothesis, typically more accurate than any of the original experts.

The learning set-up is defined as follows. We assume a fixed kernel function $K : \mathcal{X} \times \mathcal{X} \to \mathbb{R}$ that can be used to generate the entries of a kernel matrix $\mathbf{K}$. The learner receives a sample $S$ of $mp$ columns randomly selected from matrix $\mathbf{K}$ uniformly without replacement. $S$ is decomposed into $p$ subsamples $S_1, \ldots, S_p$. Each subsample $S_r$, $r \in [1, p]$, contains $m$ columns and is used to define a rank-$k$ Nyström approximation $\widetilde{\mathbf{K}}_r$. Dropping the rank subscript $k$ in favor of the sample index $r$, $\widetilde{\mathbf{K}}_r$ can be written as $\widetilde{\mathbf{K}}_r = \mathbf{C}_r \mathbf{W}_r^+ \mathbf{C}_r^\top$, where $\mathbf{C}_r$ and $\mathbf{W}_r$ denote the matrices formed from the columns of $S_r$ and $\mathbf{W}_r^+$ is the pseudo-inverse of the rank-$k$ approximation of $\mathbf{W}_r$. The learner further receives a sample $V$ of $s$ columns used to determine the weight $\mu_r \in \mathbb{R}$ attributed to each expert $\widetilde{\mathbf{K}}_r$. Thus, the general form of the approximation of $\mathbf{K}$ generated by the ensemble Nyström algorithm is

$$\widetilde{\mathbf{K}}^{ens} = \sum_{r=1}^p \mu_r \widetilde{\mathbf{K}}_r. \tag{3}$$

The mixture weights $\mu_r$ can be defined in many ways. The most straightforward choice consists of assigning equal weight to each expert, $\mu_r = 1/p$, $r \in [1, p]$. This choice does not require the additional sample $V$, but it ignores the relative quality of each Nyström approximation. Nevertheless,

this simple *uniform method* already generates a solution superior to any one of the approximations $\widetilde{\mathbf{K}}_r$ used in the combination, as we shall see in the experimental section.

Another method, the *exponential weight method*, consists of measuring the reconstruction error $\hat{\epsilon}_r$ of each expert $\widetilde{\mathbf{K}}_r$ over the validation sample $V$ and defining the mixture weight as $\mu_r = \exp(-\eta\hat{\epsilon}_r)/Z$, where $\eta > 0$ is a parameter of the algorithm and $Z$ a normalization factor ensuring that the vector $\boldsymbol{\mu} = (\mu_1, \ldots, \mu_p)$ belongs to the simplex $\Delta$ of $\mathbb{R}^p$: $\Delta = \{\boldsymbol{\mu} \in \mathbb{R}^p \colon \boldsymbol{\mu} \geq 0 \wedge \sum_{r=1}^p \mu_r = 1\}$. The choice of the mixture weights here is similar to those used in the weighted-majority algorithm [11]. Let $\mathbf{K}_V$ denote the matrix formed by using the samples from $V$ as its columns and let $\widetilde{\mathbf{K}}_r^V$ denote the submatrix of $\widetilde{\mathbf{K}}_r$ containing the columns corresponding to the columns in $V$. The reconstruction error $\hat{\epsilon}_r = \|\widetilde{\mathbf{K}}_r^V - \mathbf{K}_V\|$ can be directly computed from these matrices.

A more general class of methods consists of using the sample $V$ to train the mixture weights $\mu_r$ to optimize a regression objective function such as the following:

$$\min_{\boldsymbol{\mu}} \ \lambda\|\boldsymbol{\mu}\|_2^2 + \|\sum_{r=1}^p \mu_r \widetilde{\mathbf{K}}_r^V - \mathbf{K}_V\|_F^2, \tag{4}$$

where $\mathbf{K}_V$ denotes the matrix formed by the columns of the samples $S$ and $V$ and $\lambda > 0$. This can be viewed as a ridge regression objective function and admits a closed form solution. We will refer to this method as the *ridge regression method*.

The total complexity of the ensemble Nyström algorithm is $O(pm^3 + pmkn + C_{\boldsymbol{\mu}})$, where $C_{\boldsymbol{\mu}}$ is the cost of computing the mixture weights, $\boldsymbol{\mu}$, used to combine the $p$ Nyström approximations. In general, the cubic term dominates the complexity since the mixture weights can be computed in constant time for the uniform method, in $O(psn)$ for the exponential weight method, or in $O(p^3 + pms)$ for the ridge regression method. Furthermore, although the ensemble Nyström algorithm requires $p$ times more space and CPU cycles than the standard Nyström method, these additional requirements are quite reasonable in practice. The space requirement is still manageable for even large-scale applications given that $p$ is typically O(1) and $m$ is usually a very small percentage of $n$ (see Section 4 for further details). In terms of CPU requirements, we note that our algorithm can be easily parallelized, as all $p$ experts can be computed simultaneously. Thus, with a cluster of $p$ machines, the running time complexity of this algorithm is nearly equal to that of the standard Nyström algorithm with $m$ samples.

## 3  Theoretical analysis

We now present a theoretical analysis of the ensemble Nyström method for which we use as tools some results previously shown by [5] and [9]. As in [9], we shall use the following generalization of McDiarmid's concentration bound to sampling without replacement [3].

**Theorem 1.** *Let $Z_1, \ldots, Z_m$ be a sequence of random variables sampled uniformly without replacement from a fixed set of $m + u$ elements $Z$, and let $\phi\colon Z^m \to \mathbb{R}$ be a symmetric function such that for all $i \in [1, m]$ and for all $z_1, \ldots, z_m \in Z$ and $z_1', \ldots, z_m' \in Z$, $|\phi(z_1, \ldots, z_m) - \phi(z_1, \ldots, z_{i-1}, z_i', z_{i+1}, \ldots, z_m)| \leq c$. Then, for all $\epsilon > 0$, the following inequality holds:*

$$\Pr\left[\phi - \mathrm{E}[\phi] \geq \epsilon\right] \leq \exp\left[\tfrac{-2\epsilon^2}{\alpha(m,u)c^2}\right], \tag{5}$$

*where $\alpha(m, u) = \frac{mu}{m+u-1/2} \ \frac{1}{1-1/(2\max\{m,u\})}$.*

We define the *selection matrix* corresponding to a sample of $m$ columns as the matrix $\mathbf{S} \in \mathbb{R}^{n \times m}$ defined by $\mathbf{S}_{ii} = 1$ if the $i$th column of $\mathbf{K}$ is among those sampled, $\mathbf{S}_{ij} = 0$ otherwise. Thus, $\mathbf{C} = \mathbf{KS}$ is the matrix formed by the columns sampled. Since $\mathbf{K}$ is SPSD, there exists $\mathbf{X} \in \mathbb{R}^{N \times n}$ such that $\mathbf{K} = \mathbf{X}^\top \mathbf{X}$. We shall denote by $\mathbf{K}_{\max}$ the maximum diagonal entry of $\mathbf{K}$, $\mathbf{K}_{\max} = \max_i \mathbf{K}_{ii}$, and by $d_{\max}^{\mathbf{K}}$ the distance $\max_{ij} \sqrt{\mathbf{K}_{ii} + \mathbf{K}_{jj} - 2\mathbf{K}_{ij}}$.

### 3.1  Error bounds for the standard Nyström method

The following theorem gives an upper bound on the norm-2 error of the Nyström approximation of the form $\|\mathbf{K} - \widehat{\mathbf{K}}\|_2/\|\mathbf{K}\|_2 \leq \|\mathbf{K} - \mathbf{K}_k\|_2/\|\mathbf{K}\|_2 + O(1/\sqrt{m})$ and an upper bound on the Frobenius

error of the Nyström approximation of the form $\|\mathbf{K} - \widetilde{\mathbf{K}}\|_F / \|\mathbf{K}\|_F \leq \|\mathbf{K} - \mathbf{K}_k\|_F / \|\mathbf{K}\|_F + O(1/m^{\frac{1}{4}})$. Note that these bounds are similar to the bounds in Theorem 3 in [9], though in this work we give new results for the spectral norm and present a tighter Lipschitz condition (9), the latter of which is needed to derive tighter bounds in Section 3.2.

**Theorem 2.** *Let $\widetilde{\mathbf{K}}$ denote the rank-$k$ Nyström approximation of $\mathbf{K}$ based on $m$ columns sampled uniformly at random without replacement from $\mathbf{K}$, and $\mathbf{K}_k$ the best rank-$k$ approximation of $\mathbf{K}$. Then, with probability at least $1 - \delta$, the following inequalities hold for any sample of size $m$:*

$$\|\mathbf{K} - \widetilde{\mathbf{K}}\|_2 \leq \|\mathbf{K} - \mathbf{K}_k\|_2 + \frac{2n}{\sqrt{m}} \mathbf{K}_{\max}\left[ 1 + \sqrt{\frac{n-m}{n-1/2}\frac{1}{\beta(m,n)}} \log \frac{1}{\delta} \, d_{\max}^{\mathbf{K}}/\mathbf{K}_{\max}^{\frac{1}{2}} \right]$$

$$\|\mathbf{K} - \widetilde{\mathbf{K}}\|_F \leq \|\mathbf{K} - \mathbf{K}_k\|_F + \left[\frac{64k}{m}\right]^{\frac{1}{4}} n\mathbf{K}_{\max}\left[ 1 + \sqrt{\frac{n-m}{n-1/2}\frac{1}{\beta(m,n)}} \log \frac{1}{\delta} \, d_{\max}^{\mathbf{K}}/\mathbf{K}_{\max}^{\frac{1}{2}} \right]^{\frac{1}{2}},$$

*where $\beta(m,n) = 1 - \frac{1}{2\max\{m,n-m\}}$.*

*Proof.* To bound the norm-2 error of the Nyström method in the scenario of sampling without replacement, we start with the following general inequality given by [5][proof of Lemma 4]:

$$\|\mathbf{K} - \widetilde{\mathbf{K}}\|_2 \leq \|\mathbf{K} - \mathbf{K}_k\|_2 + 2\|\mathbf{X}\mathbf{X}^\top - \mathbf{Z}\mathbf{Z}^\top\|_2, \tag{6}$$

where $\mathbf{Z} = \sqrt{\frac{n}{m}}\mathbf{X}\mathbf{S}$. We then apply the McDiarmid-type inequality of Theorem 1 to $\phi(\mathbf{S}) = \|\mathbf{X}\mathbf{X}^\top - \mathbf{Z}\mathbf{Z}^\top\|_2$. Let $\mathbf{S}'$ be a sampling matrix selecting the same columns as $\mathbf{S}$ except for one, and let $\mathbf{Z}'$ denote $\sqrt{\frac{n}{m}}\mathbf{X}\mathbf{S}'$. Let $\mathbf{z}$ and $\mathbf{z}'$ denote the only differing columns of $\mathbf{Z}$ and $\mathbf{Z}'$, then

$$|\phi(\mathbf{S}') - \phi(\mathbf{S})| \leq \|\mathbf{z}'\mathbf{z}'^\top - \mathbf{z}\mathbf{z}^\top\|_2 = \|(\mathbf{z}' - \mathbf{z})\mathbf{z}'^\top + \mathbf{z}(\mathbf{z}' - \mathbf{z})^\top\|_2 \tag{7}$$

$$\leq 2\|\mathbf{z}' - \mathbf{z}\|_2 \max\{\|\mathbf{z}\|_2, \|\mathbf{z}'\|_2\}. \tag{8}$$

Columns of $\mathbf{Z}$ are those of $\mathbf{X}$ scaled by $\sqrt{n/m}$. The norm of the difference of two columns of $\mathbf{X}$ can be viewed as the norm of the difference of two feature vectors associated to $K$ and thus can be bounded by $d_{\mathbf{K}}$. Similarly, the norm of a single column of $\mathbf{X}$ is bounded by $\mathbf{K}_{\max}^{\frac{1}{2}}$. This leads to the following inequality:

$$|\phi(\mathbf{S}') - \phi(\mathbf{S})| \leq \frac{2n}{m} d_{\max}^{\mathbf{K}} \mathbf{K}_{\max}^{\frac{1}{2}}. \tag{9}$$

The expectation of $\phi$ can be bounded as follows:

$$\mathrm{E}[\Phi] = \mathrm{E}[\|\mathbf{X}\mathbf{X}^\top - \mathbf{Z}\mathbf{Z}^\top\|_2] \leq \mathrm{E}[\|\mathbf{X}\mathbf{X}^\top - \mathbf{Z}\mathbf{Z}^\top\|_F] \leq \frac{n}{\sqrt{m}}\mathbf{K}_{\max}, \tag{10}$$

where the last inequality follows Corollary 2 of [9]. The inequalities (9) and (10) combined with Theorem 1 give a bound on $\|\mathbf{X}\mathbf{X}^\top - \mathbf{Z}\mathbf{Z}^\top\|_2$ and yield the statement of the theorem.

The following general inequality holds for the Frobenius error of the Nyström method [5]:

$$\|\mathbf{K} - \widetilde{\mathbf{K}}\|_F^2 \leq \|\mathbf{K} - \mathbf{K}_k\|_F^2 + \sqrt{64k}\,\|\mathbf{X}\mathbf{X}^\top - \mathbf{Z}\mathbf{Z}^\top\|_F^2 \, n\mathbf{K}_{ii}^{\max}. \tag{11}$$

Bounding the term $\|\mathbf{X}\mathbf{X}^\top - \mathbf{Z}\mathbf{Z}^\top\|_F^2$ as in the norm-2 case and using the concentration bound of Theorem 1 yields the result of the theorem. $\square$

### 3.2 Error bounds for the ensemble Nyström method

The following error bounds hold for ensemble Nyström methods based on a convex combination of Nyström approximations.

**Theorem 3.** *Let $S$ be a sample of $pm$ columns drawn uniformly at random without replacement from $\mathbf{K}$, decomposed into $p$ subsamples of size $m$, $S_1, \ldots, S_p$. For $r \in [1,p]$, let $\widetilde{\mathbf{K}}_r$ denote the rank-$k$ Nyström approximation of $\mathbf{K}$ based on the sample $S_r$, and let $\mathbf{K}_k$ denote the best rank-$k$ approximation of $\mathbf{K}$. Then, with probability at least $1 - \delta$, the following inequalities hold for any sample $S$ of size $pm$ and for any $\boldsymbol{\mu}$ in the simplex $\Delta$ and $\widetilde{\mathbf{K}}^{ens} = \sum_{r=1}^{p} \mu_r \widetilde{\mathbf{K}}_r$:*

$$\|\mathbf{K} - \widetilde{\mathbf{K}}^{ens}\|_2 \leq \|\mathbf{K} - \mathbf{K}_k\|_2 + \frac{2n}{\sqrt{m}}\mathbf{K}_{\max}\left[ 1 + \mu_{\max} p^{\frac{1}{2}}\sqrt{\frac{n-pm}{n-1/2}\frac{1}{\beta(pm,n)}} \log \frac{1}{\delta} \, d_{\max}^{\mathbf{K}}/\mathbf{K}_{\max}^{\frac{1}{2}} \right]$$

$$\|\mathbf{K} - \widetilde{\mathbf{K}}^{ens}\|_F \leq \|\mathbf{K} - \mathbf{K}_k\|_F + \left[\frac{64k}{m}\right]^{\frac{1}{4}} n\mathbf{K}_{\max}\left[ 1 + \mu_{\max} p^{\frac{1}{2}}\sqrt{\frac{n-pm}{n-1/2}\frac{1}{\beta(pm,n)}} \log \frac{1}{\delta} \, d_{\max}^{\mathbf{K}}/\mathbf{K}_{\max}^{\frac{1}{2}} \right]^{\frac{1}{2}},$$

*where $\beta(pm,n) = 1 - \frac{1}{2\max\{pm,n-pm\}}$ and $\mu_{\max} = \max_{r=1}^{p} \mu_r$.*

*Proof.* For $r \in [1, p]$, let $\mathbf{Z}_r = \sqrt{n/m} \, \mathbf{X} \mathbf{S}_r$, where $\mathbf{S}_r$ denotes the selection matrix corresponding to the sample $S_r$. By definition of $\widetilde{\mathbf{K}}^{ens}$ and the upper bound on $\|\mathbf{K} - \widetilde{\mathbf{K}}_r\|_2$ already used in the proof of theorem 2, the following holds:

$$\|\mathbf{K} - \widetilde{\mathbf{K}}^{ens}\|_2 = \left\| \sum_{r=1}^{p} \mu_r (\mathbf{K} - \widetilde{\mathbf{K}}_r) \right\|_2 \leq \sum_{r=1}^{p} \mu_r \|\mathbf{K} - \widetilde{\mathbf{K}}_r\|_2 \tag{12}$$

$$\leq \sum_{r=1}^{p} \mu_r \left( \|\mathbf{K} - \mathbf{K}_k\|_2 + 2\|\mathbf{X}\mathbf{X}^\top - \mathbf{Z}_r \mathbf{Z}_r^\top\|_2 \right) \tag{13}$$

$$= \|\mathbf{K} - \mathbf{K}_k\|_2 + 2 \sum_{r=1}^{p} \mu_r \|\mathbf{X}\mathbf{X}^\top - \mathbf{Z}_r \mathbf{Z}_r^\top\|_2. \tag{14}$$

We apply Theorem 1 to $\phi(S) = \sum_{r=1}^{p} \mu_r \|\mathbf{X}\mathbf{X}^\top - \mathbf{Z}_r \mathbf{Z}_r^\top\|_2$. Let $S'$ be a sample differing from $S$ by only one column. Observe that changing one column of the full sample $S$ changes only one subsample $S_r$ and thus only one term $\mu_r \|\mathbf{X}\mathbf{X}^\top - \mathbf{Z}_r \mathbf{Z}_r^\top\|_2$. Thus, in view of the bound (9) on the change to $\|\mathbf{X}\mathbf{X}^\top - \mathbf{Z}_r \mathbf{Z}_r^\top\|_2$, the following holds:

$$|\phi(S') - \phi(S)| \leq \frac{2n}{m} \mu_{\max} d_{\max}^{\mathbf{K}} \mathbf{K}_{\max}^{\frac{1}{2}}, \tag{15}$$

The expectation of $\Phi$ can be straightforwardly bounded by $\mathrm{E}[\Phi(S)] = \sum_{r=1}^{p} \mu_r \, \mathrm{E}[\|\mathbf{X}\mathbf{X}^\top - \mathbf{Z}_r \mathbf{Z}_r^\top\|_2] \leq \sum_{r=1}^{p} \mu_r \frac{n}{\sqrt{m}} \mathbf{K}_{\max} = \frac{n}{\sqrt{m}} \mathbf{K}_{\max}$ using the bound (10) for a single expert. Plugging in this upper bound and the Lipschitz bound (15) in Theorem 1 yields our norm-2 bound for the ensemble Nyström method.

For the Frobenius error bound, using the convexity of the Frobenius norm square $\|\cdot\|_F^2$ and the general inequality (11), we can write

$$\|\mathbf{K} - \widetilde{\mathbf{K}}^{ens}\|_F^2 = \left\| \sum_{r=1}^{p} \mu_r (\mathbf{K} - \widetilde{\mathbf{K}}_r) \right\|_F^2 \leq \sum_{r=1}^{p} \mu_r \|\mathbf{K} - \widetilde{\mathbf{K}}_r\|_F^2 \tag{16}$$

$$\leq \sum_{r=1}^{p} \mu_r \left[ \|\mathbf{K} - \mathbf{K}_k\|_F^2 + \sqrt{64k} \, \|\mathbf{X}\mathbf{X}^\top - \mathbf{Z}_r \mathbf{Z}_r^\top\|_F \, n \mathbf{K}_{ii}^{\max} \right]. \tag{17}$$

$$= \|\mathbf{K} - \mathbf{K}_k\|_F^2 + \sqrt{64k} \sum_{r=1}^{p} \mu_r \|\mathbf{X}\mathbf{X}^\top - \mathbf{Z}_r \mathbf{Z}_r^\top\|_F \, n \mathbf{K}_{ii}^{\max}. \tag{18}$$

The result follows by the application of Theorem 1 to $\psi(S) = \sum_{r=1}^{p} \mu_r \|\mathbf{X}\mathbf{X}^\top - \mathbf{Z}_r \mathbf{Z}_r^\top\|_F$ in a way similar to the norm-2 case. $\qquad\square$

The bounds of Theorem 3 are similar in form to those of Theorem 2. However, the bounds for the ensemble Nyström are tighter than those for any Nyström expert based on a single sample of size $m$ even for a uniform weighting. In particular, for $\mu = 1/p$, the last term of the ensemble bound for norm-2 is smaller by a factor larger than $\mu_{\max} p^{\frac{1}{2}} = 1/\sqrt{p}$.

## 4    Experiments

In this section, we present experimental results that illustrate the performance of the ensemble Nyström method. We work with the datasets listed in Table 1. In Section 4.1, we compare the performance of various methods for calculating the mixture weights ($\mu_r$). In Section 4.2, we show the effectiveness of our technique on *large-scale* datasets. Throughout our experiments, we measure the accuracy of a low-rank approximation $\widetilde{\mathbf{K}}$ by calculating the relative error in Frobenius and spectral norms, that is, if we let $\xi = \{2, F\}$, then we calculate the following quantity:

$$\% \, \text{error} = \frac{\|\mathbf{K} - \widetilde{\mathbf{K}}\|_\xi}{\|\mathbf{K}\|_\xi} \times 100. \tag{19}$$

| Dataset | Type of data | # Points ($n$) | # Features ($d$) | Kernel |
|---|---|---|---|---|
| PIE-2.7K [16] | face images | 2731 | 2304 | linear |
| MNIST [10] | digit images | 4000 | 784 | linear |
| ESS [8] | proteins | 4728 | 16 | RBF |
| AB-S [1] | abalones | 4177 | 8 | RBF |
| DEXT [1] | bag of words | 2000 | 20000 | linear |
| SIFT-1M [12] | Image features | 1M | 128 | RBF |

Table 1: A summary of the datasets used in the experiments.

## 4.1 Ensemble Nyström with various mixture weights

In this set of experiments, we show results for our ensemble Nyström method using different techniques to choose the mixture weights as discussed in Section 2.2. We first experimented with the first five datasets shown in Table 1. For each dataset, we fixed the reduced rank to $k = 50$, and set the number of sampled columns to $m = 3\% \, n$.[1] Furthermore, for the exponential and the ridge regression variants, we sampled an additional set of $s = 20$ columns and used an additional 20 columns ($s'$) as a hold-out set for selecting the optimal values of $\eta$ and $\lambda$. The number of approximations, $p$, was varied from 2 to 30. As a baseline, we also measured the minimal and mean percent error across the $p$ Nyström approximations used to construct $\widetilde{\mathbf{K}}^{ens}$. For the Frobenius norm, we also calculated the performance when using the optimal $\boldsymbol{\mu}$, that is, we used least-square regression to find the best possible choice of combination weights for a fixed set of $p$ approximations by setting $s = n$.

The results of these experiments are presented in Figure 1 for the Frobenius norm and in Figure 2 for the spectral norm. These results clearly show that the ensemble Nyström performance is significantly better than any of the individual Nyström approximations. Furthermore, the ridge regression technique is the best of the proposed techniques and generates nearly the optimal solution in terms of the percent error in Frobenius norm. We also observed that when $s$ is increased to approximately $5\%$ to $10\%$ of $n$, linear regression without any regularization performs about as well as ridge regression for both the Frobenius and spectral norm. Figure 3 shows this comparison between linear regression and ridge regression for varying values of $s$ using a fixed number of experts ($p = 10$). Finally we note that the ensemble Nyström method tends to converge very quickly, and the most significant gain in performance occurs as $p$ increases from 2 to 10.

## 4.2 Large-scale experiments

Next, we present an empirical study of the effectiveness of the ensemble Nyström method on the SIFT-1M dataset in Table 1 containing 1 *million* data points. As is common practice with large-scale datasets, we worked on a cluster of several machines for this dataset. We present results comparing the performance of the ensemble Nyström method, using both uniform and ridge regression mixture weights, with that of the best and mean performance across the $p$ Nyström approximations used to construct $\widetilde{\mathbf{K}}^{ens}$. We also make comparisons with a recently proposed $k$-means based sampling technique for the Nyström method [19]. Although the $k$-means technique is quite effective at generating informative columns by exploiting the data distribution, the cost of performing $k$-means becomes expensive for even moderately sized datasets, making it difficult to use in large-scale settings. Nevertheless, in this work, we include the $k$-means method in our comparison, and we present results for various subsamples of the SIFT-1M dataset, with $n$ ranging from 5K to 1M.

To fairly compare these techniques, we performed 'fixed-time' experiments. To do this, we first searched for an appropriate $m$ such that the percent error for the ensemble Nyström method with ridge weights was approximately $10\%$, and measured the time required by the cluster to construct this approximation. We then alloted an equal amount of time (within 1 second) for the other techniques, and measured the quality of the resulting approximations. For these experiments, we set $k = 50$ and $p = 10$, based on the results from the previous section. Furthermore, in order to speed up computation on this large dataset, we decreased the size of the validation and hold-out sets to $s = 2$ and $s' = 2$, respectively.

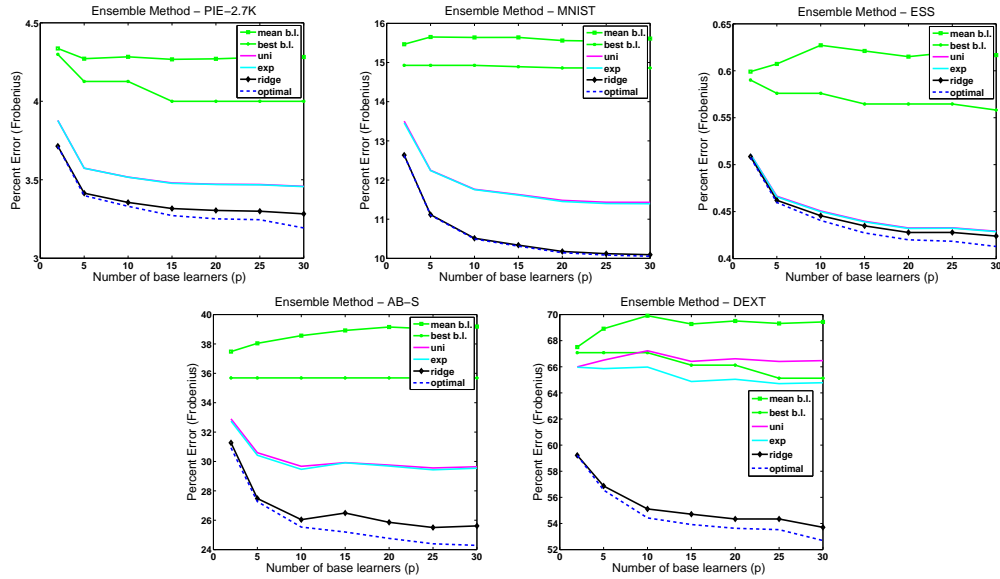

Figure 1: Percent error in Frobenius norm for ensemble Nyström method using uniform ('uni'), exponential ('exp'), ridge ('ridge') and optimal ('optimal') mixture weights as well as the best ('best b.l.') and mean ('mean b.l.') performance of the $p$ base learners used to create the ensemble approximation.

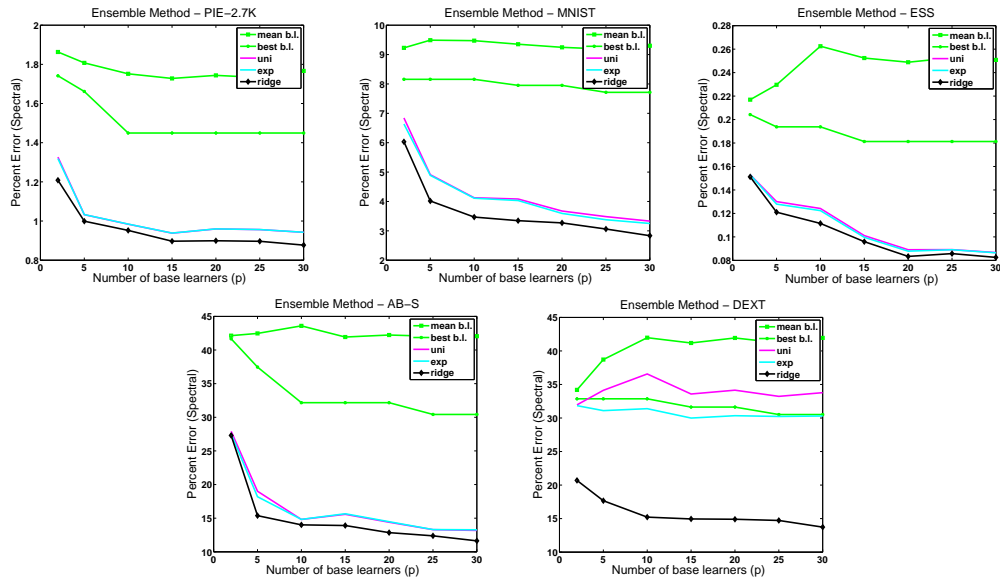

Figure 2: Percent error in spectral norm for ensemble Nyström method using various mixture weights as well as the best and mean performance of the $p$ approximations used to create the ensemble approximation. Legend entries are the same as in Figure 1.

The results of this experiment, presented in Figure 4, clearly show that the ensemble Nyström method is the most effective technique given a fixed amount of time. Furthermore, even with the small values of $s$ and $s'$, ensemble Nyström with ridge-regression weighting outperforms the uniform ensemble Nyström method. We also observe that due to the high computational cost of $k$-means for large datasets, the $k$-means approximation does not perform well in this 'fixed-time' experiment. It generates an approximation that is worse than the mean standard Nyström approximation and its performance increasingly deteriorates as $n$ approaches 1M. Finally, we note that al-

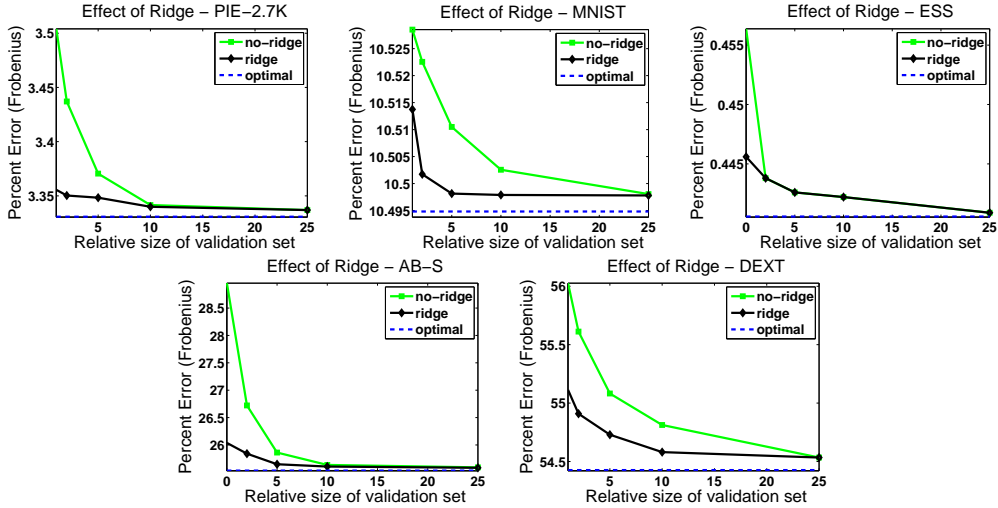

Figure 3: Comparison of percent error in Frobenius norm for the ensemble Nyström method with $p = 10$ experts with weights derived from linear regression ('no-ridge') and ridge regression ('ridge'). The dotted line indicates the optimal combination. The relative size of the validation set equals $s/n \times 100\%$.

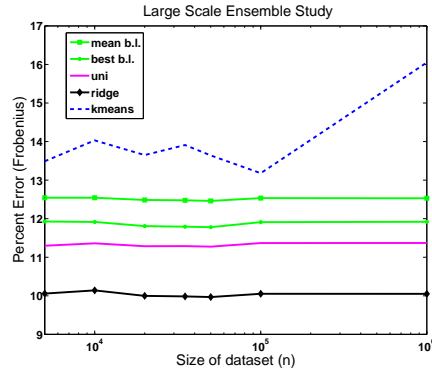

Figure 4: Large-scale performance comparison with SIFT-1M dataset. Given fixed computational time, ensemble Nyström with ridge weights tends to outperform other techniques.

though the space requirements are 10 times greater for ensemble Nyström in comparison to standard Nyström (since $p = 10$ in this experiment), the space constraints are nonetheless quite reasonable. For instance, when working with the full 1M points, the ensemble Nyström method with ridge regression weights only required approximately $1\%$ of the columns of $\mathbf{K}$ to achieve a percent error of $10\%$.

## 5 Conclusion

We presented a novel family of algorithms, *ensemble Nyström algorithms*, for accurate low-rank approximations in large-scale applications. The consistent and significant performance improvement across a number of different data sets, along with the fact that these algorithms can be easily parallelized, suggests that these algorithms can benefit a variety of applications where kernel methods are used. Interestingly, the algorithmic solution we have proposed for scaling these kernel learning algorithms to larger scales is itself derived from the machine learning idea of ensemble methods. We also gave the first theoretical analysis of these methods. We expect that finer error bounds and theoretical guarantees will further guide the design of the ensemble algorithms and help us gain a better insight about the convergence properties of our algorithms.

## Footnotes

[1] Similar results (not reported here) were observed for other values of $k$ and $m$ as well.

# References

[1] A. Asuncion and D. Newman. UCI machine learning repository, 2007.

[2] B. E. Boser, I. Guyon, and V. N. Vapnik. A training algorithm for optimal margin classifiers. In *COLT*, volume 5, pages 144–152, 1992.

[3] C. Cortes, M. Mohri, D. Pechyony, and A. Rastogi. Stability of transductive regression algorithms. In *ICML*, 2008.

[4] C. Cortes and V. N. Vapnik. Support-Vector Networks. *Machine Learning*, 20(3):273–297, 1995.

[5] P. Drineas and M. W. Mahoney. On the Nyström method for approximating a Gram matrix for improved kernel-based learning. *JMLR*, 6:2153–2175, 2005.

[6] C. Fowlkes, S. Belongie, F. Chung, and J. Malik. Spectral grouping using the Nyström method. *IEEE Transactions on Pattern Analysis and Machine Intelligence*, 26(2), 2004.

[7] G. Golub and C. V. Loan. *Matrix Computations*. Johns Hopkins University Press, Baltimore, 2nd edition, 1983.

[8] A. Gustafson, E. Snitkin, S. Parker, C. DeLisi, and S. Kasif. Towards the identification of essential genes using targeted genome sequencing and comparative analysis. *BMC:Genomics*, 7:265, 2006.

[9] S. Kumar, M. Mohri, and A. Talwalkar. Sampling techniques for the Nyström method. In *AISTATS*, pages 304–311, 2009.

[10] Y. LeCun and C. Cortes. The MNIST database of handwritten digits, 2009.

[11] N. Littlestone and M. K. Warmuth. The weighted majority algorithm. *Information and Computation*, 108(2):212261, 1994.

[12] D. G. Lowe. Distinctive image features from scale-invariant keypoints. *International Journal of Computer Vision*, 60:91–110, 2004.

[13] J. C. Platt. Fast embedding of sparse similarity graphs. In *NIPS*, 2004.

[14] C. Saunders, A. Gammerman, and V. Vovk. Ridge Regression Learning Algorithm in Dual Variables. In *Proceedings of the ICML '98*, pages 515–521, 1998.

[15] B. Schölkopf, A. Smola, and K.-R. Müller. Nonlinear component analysis as a kernel eigenvalue problem. *Neural Computation*, 10(5):1299–1319, 1998.

[16] T. Sim, S. Baker, and M. Bsat. The CMU PIE database. In *Conference on Automatic Face and Gesture Recognition*, 2002.

[17] A. Talwalkar, S. Kumar, and H. Rowley. Large-scale manifold learning. In *CVPR*, 2008.

[18] C. K. I. Williams and M. Seeger. Using the Nyström method to speed up kernel machines. In *NIPS*, pages 682–688, 2000.

[19] K. Zhang, I. Tsang, and J. Kwok. Improved Nyström low-rank approximation and error analysis. In *ICML*, pages 273–297, 2008.

